# Evaluation of Adaptive Mixtures of Competing Experts

**Steven J. Nowlan** and **Geoffrey E. Hinton**
Computer Science Dept.
University of Toronto
Toronto, ONT M5S 1A4

## Abstract

We compare the performance of the modular architecture, composed of competing expert networks, suggested by Jacobs, Jordan, Nowlan and Hinton (1991) to the performance of a single back-propagation network on a complex, but low-dimensional, vowel recognition task. Simulations reveal that this system is capable of uncovering interesting decompositions in a complex task. The type of decomposition is strongly influenced by the nature of the input to the gating network that decides which expert to use for each case. The modular architecture also exhibits consistently better generalization on many variations of the task.

## 1 Introduction

If back-propagation is used to train a single, multilayer network to perform different subtasks on different occasions, there will generally be strong interference effects which lead to slow learning and poor generalization. If we know in advance that a set of training cases may be naturally divided into subsets that correspond to distinct subtasks, interference can be reduced by using a system (see Fig. 1) composed of several different "expert" networks plus a gating network that decides which of the experts should be used for each training case.

Systems of this type have been suggested by a number of authors (Hampshire and Waibel, 1989; Jacobs, Jordan and Barto, 1990; Jacobs et al., 1991) (see also the paper by Jacobs and Jordan in this volume (1991)). Jacobs, Jordan, Nowlan and Hinton (1991) show that this system can be trained by performing gradient descent

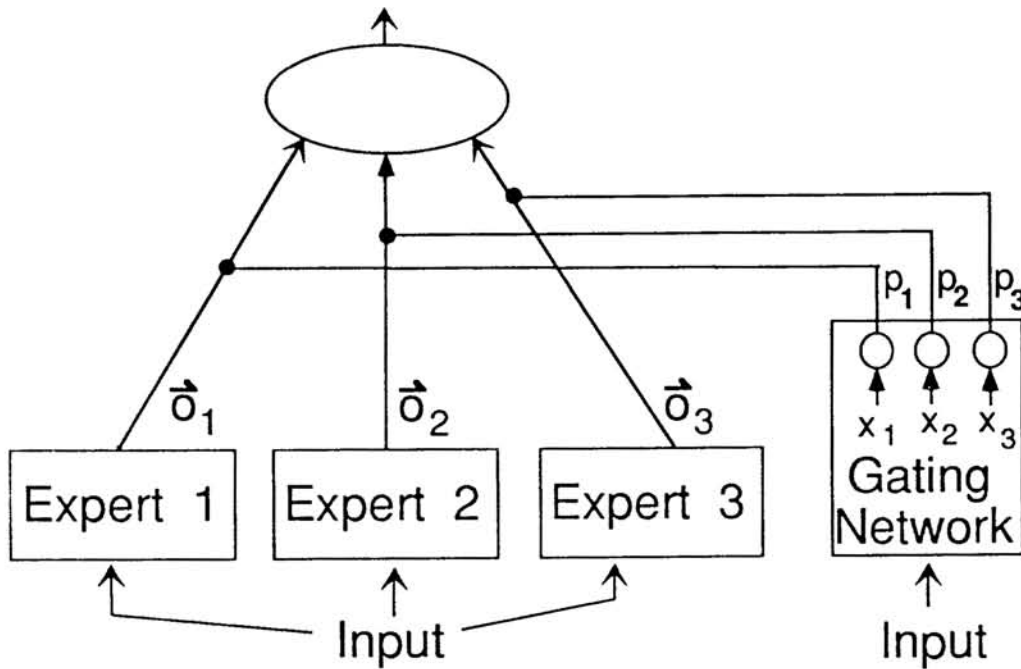

Figure 1: A system of expert and gating networks. Each expert is a feedforward network and all experts receive the same input and have the same number of outputs. The gating network is also feedforward and may receive a different input than the expert networks. It has normalized outputs $p_j = \exp(x_j)/\sum_i \exp(x_i)$, where $x_j$ is the total weighted input received by output unit $j$ of the gating network. $p_j$ can be viewed as the probability of selecting expert $j$ for a particular case.

in the following error function:

$$E^c = -\log \sum_i p_i^c e^{-\|\vec{d}^c - \vec{o}_i^c\|^2 / 2\sigma^2} \tag{1}$$

where $E^c$ is the error on training case $c$, $p_i^c$ is the output of the gating network for expert $i$, $\vec{d}^c$ is the desired output vector and $\vec{o}_i^c$ is the output vector of expert $i$, and $\sigma$ is constant.

The error defined by Equation 1 is simply the negative log probability of generating the desired output vector under a mixture of gaussians model of the probability distribution of possible output vectors given the current input. The output vector of each expert specifies the mean of a multidimensional gaussian distribution. These means are a function of the inputs to the experts. The outputs of the gating network specify the mixing proportions of the experts, so these too are determined by the current input.

During learning, the gradient descent in $E$ has two effects. It raises the mixing proportion of experts that do better than average in predicting the desired output vector for a particular case, and it also makes each expert better at predicting the desired output for those cases for which it has a high mixing proportion. The result of these two effects is that, after learning, the gating network nearly always assigns a mixing proportion near 1 to one expert on each case. So towards the end of the learning, each expert can focus on modelling the cases it is good at without interference from the cases for which it has a negligible mixing proportion.

In this paper, we compare mixtures of experts to single back-propagation networks on a vowel recognition task. We demonstrate that the mixtures are better at fitting the training data and better at generalizing than comparable single back-propagation networks.

## 2    Data and Experimental Procedures

The data used in these experiments consisted of the frequencies of the first and second formants for 10 vowels from 75 speakers (32 Males, 28 Females, and 15 Children) (Peterson and Barney, 1952).[1] The vowels, which were uttered in an **hVd** context, were {*heed, hid, head, had, hud, hod, hawed, hood, who'd, heard*}. The word list was repeated twice by each speaker, with the words in a different random order for each presentation. The resulting spectrograms were hand segmented and the frequencies of the formants extracted from the middle portion of the vowel.

The simulations were performed using a conjugate gradient technique, with one weight change after each pass through the training set. For the back-propagation experiments, each simulation was initialised randomly with weight values in the range $[-0.5, 0.5]$. For the mixture systems, the last layer of weights in the gating network was always initialised to 0 so that all experts initially had equal *a priori* selection probabilities, $p_{i,k}$, while all other weights in the gating and expert networks were initialized randomly with values in the range $[-0.5, 0.5]$ to break symmetry. The value of $\sigma$ used was 0.25 for all of the mixture simulations. In all cases, the input formant values were linearly scaled by dividing them by 1000, so the first formant was in the range $(0, 1.5)$ and the second was in the range $(0, 4)$.

Two sets of experiments were performed: one in which the performance of different systems on the training data was compared and a second in which the ability of different systems to generalize was compared.

Five different types of input were used in each set of experiments:

1. Frequencies of first and second formants only (Form.).

2. Form. plus a localist encoding of the speaker identity (Form. + Speaker ID).

3. Form. plus a localist encoding of whether the speaker was a male, female, or child (Form. + MFC).

4. Form. plus the minimum and maximum frequency for the first and second formant (as real values) over all samples from the speaker (Form. + Range).

5. Form. + MFC + Range.

For the simulations in which a single back-propagation network was used the network received the entire set of input values. However, for the mixture systems the expert networks saw *only* the formant frequencies, while the gating network saw everything *but* the formant frequencies (except of course when the input consisted only of the formant frequencies).

| Type of Input | # Experts | # Hid per Expert | # Hid Gating |
|---|---|---|---|
| Form. | 20 | 3 - 5 | 10 |
| Form. + Speaker ID | 10 | 25 | 0 |
| Form. + MFC | 10 | 25 | 0 |
| Form. + MFC + Range | 10 | 25 | 5 |
| Form. + Range | 10 | 25 | 5 |

Table 1: Summary of mixture architecture used with each type of input.

| Type of Input | Mixture Error % | BP Error % | Sig.(p) |
|---|---|---|---|
| Formants only | 13.9 ± 0.9 | 21.8 ± 0.6 | ≫ 0.9999 |
| Form. + Speaker ID | 4.6 ± 0.7 | 6.2 ± 0.6 | > 0.97 |
| Form. + MFC | 13.0 ± 0.4 | 15.4 ± 0.3 | ≫ 0.9999 |
| Form. + MFC + Range | 5.6 ± 0.6 | 13.1 ± 1.0 | ≫ 0.9999 |
| Form. + Range | 11.6 ± 0.9 | 13.5 ± 0.4 | > 0.998 |

Table 2: Performance comparison of associative mixture systems and single back-propagation networks on vowel classification task. Results reported are based on an average over 25 simulations for each back-propagation network or mixture system.

The BP networks used in the single network simulations contained one layer of hidden units.[2] In the mixture systems, the expert networks also contained one layer of hidden units although the number of hidden units in each expert varied. The gating network in some cases contained hidden units, while in other cases it did not (see Table 1). Further details of the simulations may be found in (Nowlan, 1991).

## 3    Results of Performance Studies

In the set of *performance* experiments, each system was trained with the entire set of 1494 tokens until the magnitude of the gradient vector was $< 10^{-8}$. The error rate (as percent of total cases) was evaluated on the *training* data (generalization studies are described in the next section). The very high degree of class overlap in this task makes it extremely difficult to find good solutions with a gradient descent procedure and this is reflected by the far from optimal average performance of all systems on the training data (see Table 2). For purposes of comparison, the best performance ever obtained on this vowel data using *speaker dependant* classification methods is about 2.5% (Gerstman, 1968; Watrous, 1990).

Table 2 reveals that in every case the mixture system performs significantly better[3] than a single network given the same input. The most striking, and interesting,

| Spec. # | % Male | % Female | % Child | % Total |
|---|---|---|---|---|
| 0 | 0.0 | 0.0 | 6.7 | 1.3 |
| 4 | 3.1 | 3.6 | 0.0 | 2.7 |
| 5 | 84.4 | 17.8 | 0.0 | 42.7 |
| 7 | 9.4 | 7.1 | 6.7 | 8.0 |
| 8 | 3.1 | 42.9 | 0.0 | 17.3 |
| 9 | 0.0 | 28.6 | 86.7 | 28.0 |

Table 3: Speaker decomposition in terms of Male, Female and Child categories for a mixture with speaker identity as input to the gating network.

result in Table 2 is contained in the fourth row of the table. While the associative mixture architecture is able to combine the two separate cues of MFC categories and speaker formant range quite effectively, the single back-propagation network fails to do so. The combination of these two different cues in the associative mixture system was obtained by a hierarchical training procedure in which three different experts were first created using the MFC cue alone, and copies of these networks were further specialized when the formant range cue was added to the input received by the gating network (see (Nowlan, 1990; Nowlan, 1991) for details). Since the single back-propagation network is much less modular than the associative mixture system, it is difficult to implement such a hierarchical training procedure in the single network case. (A variety of techniques were explored and details may again be found in (Nowlan, 1991).)

Another interesting aspect of the mixture systems, not revealed in Table 2, is the manner in which the training cases were divided among the different expert networks. Once the network was trained, the training cases were clustered by assigning each case to the expert that was selected most strongly by the gating network.

The mixture which used only the formant frequencies as input to both the gating and expert networks tended to cluster training cases according to the position of the tongue hump when the vowel is uttered. In all simulations, the four *front* vowels were always clustered together and handled by a single expert. The *low back* and *high back* vowels also tended to be grouped together, but each of these groups was divided among several experts and not always in exactly the same way.

The mixture which received speaker identity as well as formant frequencies as input tended to group speakers roughly according to the categories male, female, and child. A typical grouping of speakers by the mixture is shown in Table 3.

## 4   Results of Generalization Studies

In the set of *generalization* experiments, for all but the input which contained the speaker identity, each system was trained on data from 65 speakers until the magnitude of the gradient vector was $< 10^{-4}$. The performance was then tested on the data from the 10 speakers not in the training set. Twenty different test sets were created by leaving out different speakers for each, and results are an average over one simulation with each of the test sets. Each test set consisted of 4 male, 3

| Type of Input | Mixture Error % | BP Error % | Sig.(p) |
|---|---|---|---|
| Formants only | $15.1 \pm 0.9$ | $23.3 \pm 1.2$ | $\gg 0.9999$ |
| Form. + Speaker ID | $6.4 \pm 1.3$ | – | $\gg 0.9999$ |
| Form. + MFC | $13.5 \pm 0.6$ | $18.4 \pm 1.1$ | $\gg 0.9999$ |
| Form. + MFC + Range | $6.2 \pm 0.9$ | $16.1 \pm 1.0$ | $\gg 0.9999$ |
| Form. + Range | $12.8 \pm 0.9$ | $16.2 \pm 0.8$ | $> 0.9999$ |

Table 4: Generalization comparison of associative mixture systems and single back-propagation networks on vowel classification task. Results reported are based on an average over 20 simulations for each back-propagation network or mixture system.

female and 3 child speakers.

The generalization tests for the mixture in which speaker identity was part of the input used a different testing strategy. In this case, the training set consisted of 70 speakers and the testing set contained the remaining 5 speakers (2 male, 2 female, 1 child). Again, results are averaged over 20 different testing sets. After the mixture was trained, an expert was selected for each test speaker using one utterance of each of the first 3 vowels, and the performance of the selected expert was tested on the remaining 17 utterances of that speaker. No generalization results are reported for the single back-propagation network which received the speaker identity as well as the first and second formant values, since there is no straightforward way to perform rapid speaker adaptation with this architecture. (See Watrous (Watrous, 1990) for some approaches to speaker adaptation in single networks.) The percentage of misclassifications on the *test* set for the mixture systems and corresponding single back-propagation networks are summarized in Table 4, and in all cases the mixture system generalizes significantly better[4] than a single network.

The relatively poor generalization performance of the single back-propagation networks is not due to overfitting on the training data because the single back-propagation networks perform worse on the *training* data than the mixture systems on the *test* data. Also, the associative mixture systems initially contained even more parameters than the corresponding back-propagation networks. (The associative mixture which received formant range data for gating input initially contained almost 3600 parameters, while the corresponding single back-propagation network contained only slightly more than 1200 parameters.) Part of the explanation for the good generalization performance of the mixtures is the pruning of excess parameters as the system is trained. The number of effective parameters in the final mixture is very often less than half the number in the original system, because a large number of experts have negligible mixing proportions in the final mixture.

## 5   Discussion

The mixture systems outperform single back-propagation networks which receive the same input, and show much better generalization properties when forced to deal with relatively small training sets. In addition, the mixtures can easily be

refined hierarchically by learning a few experts and then making several copies of each and adding additional contextual input to the gating network.

The best performance for either single networks or mixture systems is obtained by including the speaker identity as part of the input. When given such input, the mixture systems are capable of discovering speaker categories which give levels of classification performance close to those obtained by speaker dependent classification schemes. Good performance can also be obtained on novel speakers by determining which existing speaker category the new speaker is most similar to (using a small number of labelled utterances). If, instead, the speaker is represented in terms of features such as male, female, child, and formant range, the mixtures also exhibit good generalization to novel speakers described in terms of these features.

## Acknowledgements

This research was supported by grants from the Natural Sciences and Engineering Research Council, the Ontario Information Technology Research Center, and Apple Computer Inc. Hinton is the Noranda fellow of the Canadian Institute for Advanced Research.

## Footnotes

[1]Obtained, with thanks, from Ray Watrous, who originally obtained the data from Ann Syrdal at AT&T Bell Labs.

[2]The number of hidden units was selected by performing a number of initial simulations with different numbers of hidden units for each network and choosing the smallest number which gave near optimal performance. These numbers were 50, 150, 60, 150, and 80 respectively for the five types of input listed above.

[3]Based on a t-test with 48 degrees of freedom.

[4]Based on a *t*-test with 38 degrees of freedom.

# References

Gerstman, L. J. (1968). Classification of self-normalized vowels. *IEEE Trans. on Audio and Electroacoustics*, AU-16(1):78–80.

Hampshire, J. and Waibel, A. (1989). The Meta-Pi network: Building distributed knowledge representations for robust pattern recognition. Technical Report CMU-CS-89-166, Carnegie-Mellon, Pittsburgh, PA.

Jacobs, R. A. and Jordan, M. I. (1991). A competitive modular connectionist architecture. In Touretzky, D. S., editor, *Neural Information Processing Systems 3*. Morgan Kauffman, San Mateo, CA.

Jacobs, R. A., Jordan, M. I., and Barto, A. G. (1990). Task decomposition through competition in a modular connectionist architecture: The what and where vision tasks. *Cognitive Science*. In Press.

Jacobs, R. A., Jordan, M. I., Nowlan, S. J., and Hinton, G. E. (1991). Adaptive mixtures of local experts. *Neural Computation*, 3(1).

Nowlan, S. J. (1990). Competing experts: An experimental investigation of assso-ciative mixture models. Technical Report CRG-TR-90-5, Department of Computer Science, University of Toronto.

Nowlan, S. J. (1991). *Soft Competitive Adaptation: Neural Network Learning Algorithms based on Fitting Statistical Mixtures*. PhD thesis, School of Computer Science, Carnegie Mellon University, Pittsburgh, PA.

Peterson, G. E. and Barney, H. L. (1952). Control methods used in a study of vowels. *The Journal of the Acoustical Society of America*, 24:175–184.

Watrous, R. L. (1990). Speaker normalization and adaptation using second order connectionist networks. Technical Report CRG-TR-90-6, University of Toronto.